# Dual-Perspective Activation: Efficient Channel Denoising via Joint Forward-Backward Criterion for Artificial Neural Networks

**Tian Qiu** [1]    **Chenchao Gao** [3]    **Zunlei Feng** [1,2*]    **Jie Lei** [4]    **Bingde Hu** [1,5]
**Xingen Wang** [1,5]    **Yi Gao** [1]    **Mingli Song** [1,2]

[1] State Key Laboratory of Blockchain and Data Security, Zhejiang University
[2] Hangzhou High-Tech Zone (Binjiang) Institute of Blockchain and Data Security
[3] Dalian University of Technology    [4] Zhejiang University of Technology
[5] Bangsheng Technology Co., Ltd.

```
{tqiu,zunleifeng,tonyhu,newroot,gaoyi,songml}@zju.edu.cn
                gccgyllyp@mail.dlut.edu.cn
                   jasonlei@zjut.edu.cn
```

## Abstract

The design of Artificial Neural Network (ANN) is inspired by the working patterns of the human brain. Connections in biological neural networks are sparse, as they only exist between few neurons. Meanwhile, the sparse representation in ANNs has been shown to possess significant advantages. Activation responses of ANNs are typically expected to promote sparse representations, where key signals get activated while irrelevant/redundant signals are suppressed. It can be observed that samples of each category are only correlated with sparse and specific channels in ANNs. However, existing activation mechanisms often struggle to suppress signals from other irrelevant channels entirely, and these signals have been verified to be detrimental to the network's final decision. To address the issue of channel noise interference in ANNs, a novel end-to-end trainable Dual-Perspective Activation (DPA) mechanism is proposed. DPA efficiently identifies irrelevant channels and applies channel denoising under the guidance of a joint criterion established online from both forward and backward propagation perspectives while preserving activation responses from relevant channels. Extensive experiments demonstrate that DPA successfully denoises channels and facilitates sparser neural representations. Moreover, DPA is parameter-free, fast, applicable to many mainstream ANN architectures, and achieves remarkable performance compared to other existing activation counterparts across multiple tasks and domains. Code is available at https://github.com/horrible-dong/DPA.

## 1 Introduction

In recent years, Artificial Neural Networks (ANNs) [1] have achieved notable advancements in a wide range of computer vision tasks [2] as well as various other tasks and domains [3–6].

The ANN design is inspired by the working patterns of the human brain [7]. Multi-Layer Perceptron (MLP) [8], a classical artificial neural network that utilizes neurons as its basic unit, closely resembles biological neural networks. Another significant development in the field is the Convolutional Neural

---

[*]Corresponding author.

Networks (CNNs) [9], which were designed to mimic the local perception mechanisms in the visual cortex. With the availability of more computational power, Vision Transformers (ViTs) [10] have emerged as a prominent advancement, combining attention mechanisms and MLPs to achieve highly effective image recognition.

Connections within biological neural networks are sparse, as they only exist between a small number of neurons [11, 12]. Even in a strongly driven visual cortex, only 1.6% to 4% of neurons are activated simultaneously at any moment [12]. Meanwhile, the sparse representation in ANNs has demonstrated notable benefits to network interpretability and generalization [13, 14]. Activation responses [15] of ANNs are typically intended to encourage sparse representations, where important signals are expected to get activated while irrelevant/redundant signals be suppressed during transmission. Rectified Linear Unit (ReLU) [14] is a commonly used activation mechanism in the deep learning field, with a response threshold of zero, allowing only inputs greater than zero to pass through while suppressing all other inputs to zero. ReLU is simple and robust, which enables a network to acquire a certain sparse representation and offers advantages such as information disentanglement, linear separability, and potential generalization ability [14]. As a result, ReLU has gained widespread applications across diverse ANN architectures.

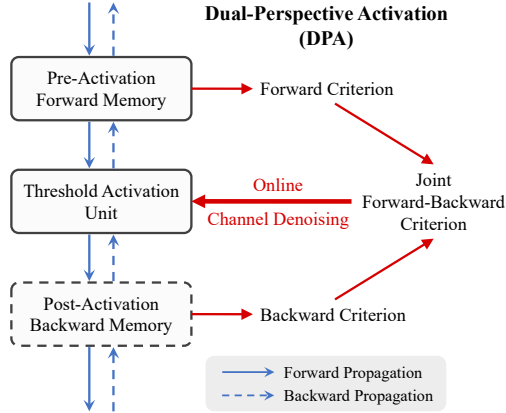

Figure 1: The proposed Dual-Perspective Activation (DPA) consists of three components: Pre-Activation Forward Memory (PreA-FM), Threshold Activation Unit (TAU), and Post-Activation Backward Memory (PostA-BM). DPA aims to efficiently identify irrelevant channels and apply channel denoising under the guidance of a joint criterion established online from both forward and backward propagation perspectives while preserving activation responses from relevant channels.

However, although the existing activation mechanisms can achieve sparsity to some extent, they remain inadequate in eliminating irrelevant/redundant features. Firstly, as depicted in Figure 2, both forward and backward propagation reveal that samples from each category are only correlated with sparse and specific channels, while a considerable number of channels are redundant, and there are significant differences in the average activation response to stimuli from different categories (also known as category specificity [16]). Additionally, as shown in Figure 3(a), the activation response distributions for each category reveal that the relevant channels mainly maintain positive responses, and ideally, the responses of irrelevant channels should be suppressed entirely; however, it can be observed in Figure 3(a) that a considerable number of responses still exist in potential irrelevant channels (indicated by red arrows). Furthermore, Figure 3(b) verifies the negative impact of the noise from these potential irrelevant channels on the network. When irrelevant channels are *manually* removed by forcing them to zero during training, the *training* accuracy significantly improves.

To address the observed deficiencies in the previous activation mechanisms regarding their limited ability to suppress noise from irrelevant channels, we propose a novel end-to-end trainable mechanism called Dual-Perspective Activation (DPA). This mechanism combines criteria established online from both forward and backward propagation perspectives to identify irrelevant channels. From the forward perspective, historical value statistics of pre-activation responses for each category are tracked online and in real time, establishing a forward criterion for channel relevance based on the threshold activation principle. From the backward perspective, historical gradient statistics of post-activation responses for each category are also tracked online and in real time, establishing a backward criterion for channel relevance based on the gradient attribution principle. The ultimate criterion is the intersection of the forward and backward criteria. Guided by this joint criterion, channel-wise denoising is conducted to suppress activation responses from irrelevant channels while preserving activation responses from relevant channels.

The contributions are summarized as follows:

- The observations on existing activation mechanisms in ANNs concerning their limited ability to suppress irrelevant features for pure sparsity, and the verification of the negative impact of noise from irrelevant channels on the network's final decision.

- The proposal of a novel end-to-end trainable mechanism, called Dual-Perspective Activation (DPA), which efficiently identifies irrelevant channels and applies channel denoising by incorporating criteria established and updated online from both forward and backward propagation perspectives while preserving activation responses from relevant channels.
- Extensive experiments have been conducted to assess the effectiveness and generalization of the proposed DPA mechanism. DPA is parameter-free and fast and achieves remarkable performance compared to existing activation counterparts across various mainstream ANN architectures and datasets, as well as multiple tasks and domains.

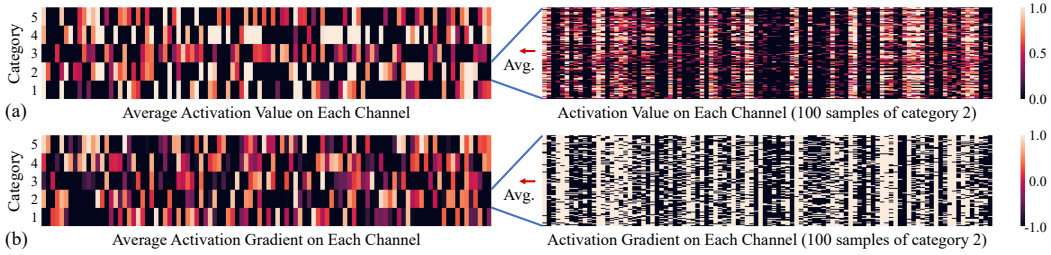

Figure 2: Category channel activation values (a) and channel activation gradients (b) are computed on the last block of ViT-Tiny. The activation layer used is ReLU. Each vector (left) is obtained by taking the average of 100 samples (right) randomly selected from its respective category. The results for five categories in CIFAR-100 are presented, with the values and gradients of the first 100 channels displayed. The horizontal axis represents the channel index, and the vertical axis represents the category index (left) / sample index (right). The brightness reflects the magnitude of the value / gradient. Here, we only focus on the sign of the gradient as the magnitude of the gradient is unstable. Therefore, the gradient of each sample is binarized to +1 and -1.

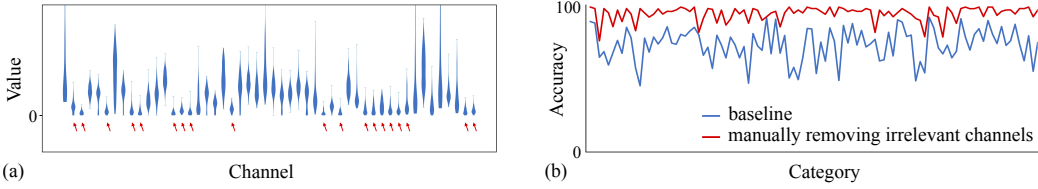

Figure 3: (a) Distributions of channel activation values in ViT-Tiny's last block are recorded by feeding samples from a given category in CIFAR-100. The activation layer used is ReLU. The horizontal axis represents the channel index, the vertical axis represents the channel activation value, and the area represents the value density. The distributions of the first 50 channels are shown. The red arrows point to potential irrelevant channels. (b) A confirmatory experiment is conducted on CIFAR-100 to compare the *training* accuracy between the original ViT-Tiny (baseline) and the ViT-Tiny with irrelevant channels *manually* removed for each category.

## 2 Observations

This section presents the observed phenomena in the activation response of ANNs and explores how these phenomena affect the network's final decision. These explorations can offer valuable insights for addressing imperfections in the activation mechanism of ANNs.

***Observation 1:*** *Each category is only correlated with sparse and specific channels in ANNs, while a considerable number of channels are redundant, and both the activation value and activation gradient show channel-wise differences between categories.*

With threshold activation, relevant features are activated while irrelevant features are suppressed, and the channel's average activation value is positively correlated with the importance of that channel. Regarding gradient attribution, the channel's average gradient is positively correlated with the importance of that channel to the prediction. The channel's activation value and activation gradient of ANN for given category samples are illustrated in Figure 2. The brighter the channel, the higher the channel relevance. Whether from the perspective of forward or backward propagation, intra-class consistency and inter-class differences can be observed. Each category is only highly correlated with sparse and specific channels, indicating that a significant proportion of channels are redundant and

ideally should not generate any responses. Additionally, the judgments from these two perspectives in opposite directions share an overlap that points to some common channels.

***Observation 2:*** *Current activation mechanisms cannot entirely eliminate responses from irrelevant channels, which have a negative impact on the network's final decision.*

As shown in Figure 3(a), potential irrelevant channels are indicated by the red arrows, and it is evident that the current activation mechanism, such as ReLU, does not entirely suppress irrelevant channels, as a considerable number of responses still persist in these channels. A confirmatory experiment is also conducted to investigate the potential impact of responses from irrelevant channels. Figure 3(b) illustrates the change in *training* accuracy when we retain only relevant channels for each category and *manually* set irrelevant ones to zero. After *manually* removing the irrelevant channels for each category, there is a substantial improvement in the *training* accuracy. This finding suggests that the responses from irrelevant channels are perceived as noise interference by the network, which adversely affects the network's final decision.

# 3 Dual-Perspective Activation

As shown in Figure 1, the proposed Dual-Perspective Activation (DPA) neuron consists of a Pre-Activation Forward Memory (PreA-FM), a Threshold Activation Unit (TAU), and a Post-Activation Backward Memory (PostA-BM). TAU processes the input signals of the network. Meanwhile, PreA-FM and PostA-BM track the historical response value and gradient before and after TAU, respectively, for each category in real time. By utilizing the memories from both PreA-FM and PostA-BM, a joint forward-backward criterion is established and updated online to measure the relevance of channels. Under the guidance of this real-time updated criterion, channel denoising is performed to suppress responses from irrelevant channels during training. The proposed DPA is end-to-end trainable, parameter-free, and fast that does not affect the inference speed.

## 3.1 Threshold Activation Unit

Threshold Activation Unit (TAU) is a basic part of the proposed DPA, simulating the activation behavior of pulse neurons [17]. Given a non-negative threshold $\tau$ and an input value $x$, if the input value $x$ exceeds the threshold $\tau$, it gets activated and propagated to the subsequent layer; otherwise, it is suppressed to zero. The TAU's operation on input $x$ can be expressed as

$$TAU(x) = \begin{cases} x, & if \ x \geq \tau \\ 0, & if \ x < \tau \end{cases}. \tag{1}$$

Generally, given a pre-activation feature map $\mathbf{F} \in \mathbb{R}^{H \times W \times C}$ with $C$ channels, $\mathbf{A} \in \mathbb{R}^{H \times W \times C}$ refers to the post-activation feature map obtained by passing $\mathbf{F}$ through the threshold activation unit TAU:

$$\mathbf{A} = TAU(\mathbf{F}). \tag{2}$$

TAU can achieve a modest level of sparsity, but its ability to further eliminate irrelevant noises is limited. Subsequently, we will enhance this basic component by introducing a denoising mechanism.

## 3.2 Joint Forward-Backward Criterion

This section introduces the construction of the criterion for channel relevance from both forward and backward propagation perspectives, based on the *Observation 1* in §2.

***From the forward propagation perspective,*** we introduce a Pre-Activation Forward Memory (PreA-FM) to store and update historical global feature value statistics.

Define the pre-activation global feature value $\mathbf{f} \in \mathbb{R}^C$ to be the globally average pooled output of $\mathbf{F}$:

$$\mathbf{f} = \frac{1}{HW} \sum_{h=1}^{H} \sum_{w=1}^{W} \mathbf{F}_{h,w}. \tag{3}$$

Let $K$ denote the number of categories of a given dataset. A PreA-FM contains $K$ forward memories $\{\boldsymbol{\mu}^k\}_{k=1}^{K}$, respectively for the $K$ categories, where $\boldsymbol{\mu}^k \in \mathbb{R}^C$ stores the historical mean value before the activation TAU. For input data of each category $k$, the pre-activation global feature value $\mathbf{f}^k$

within the network is computed and updated in the historical mean $\boldsymbol{\mu}^k$ separately and in real time. To avoid an increase in space complexity, $\boldsymbol{\mu}^k$ is updated by using a moving mean technique:

$$\boldsymbol{\mu}^{k,t} = (1-m) \cdot \boldsymbol{\mu}^{k,t-1} + m \cdot \mathbf{f}^{k,t}, \tag{4}$$

where $\mathbf{f}^{k,t}$ represents the pre-activation global feature value of a sample of category $k$ at time $t$. $\boldsymbol{\mu}^{k,t-1}$ and $\boldsymbol{\mu}^{k,t}$ represent the historical mean value of category $k$ at times $t-1$ and $t$, respectively. The hyperparameter $m$ is the momentum for updating the moving mean. Therefore, only $K$ memory vectors $\{\boldsymbol{\mu}^k\}_{k=1}^{K}$ need to be maintained throughout the entire process, whose space complexity is negligible. The statistical analysis of historical input values allows for the examination of long-term trends and patterns, providing more reliable evidence for identifying irrelevant channels.

Based on the principle of threshold activation, signals from irrelevant channels learn to be below the threshold before the activation, resulting in their being suppressed after the activation. The forward channel relevance criterion of each category denotes $\{\mathscr{R}^{k,F}\}_{k=1}^{K}$, where $F$ denotes the term "Forward" and $\mathscr{R}^{k,F} \in \mathbb{R}^C$ can be established as

$$\mathscr{R}_c^{k,F} = \begin{cases} 0, & if \;\; \boldsymbol{\mu}_c^k \geq \tau \\ 1, & if \;\; \boldsymbol{\mu}_c^k < \tau \end{cases}, \tag{5}$$

where the indicator "1" signifies potential irrelevant channels.

***From the backward propagation perspective,*** we introduce a Post-Activation Backward Memory (PostA-BM) to store and update historical global feature gradient statistics.

Define the post-activation global feature gradient $\mathbf{g} \in \mathbb{R}^C$ to be the globally average pooled output of the gradient of the prediction score $\hat{y}$ w.r.t. the activated feature map $\mathbf{A}$:

$$\mathbf{g} = \frac{1}{HW} \sum_{h=1}^{H} \sum_{w=1}^{W} \frac{\partial \hat{y}}{\partial \mathbf{A}_{h,w}}. \tag{6}$$

Here, we only focus on the sign of gradient as the magnitude of gradient is unstable. So, the gradient is binarized as follows:

$$\tilde{\mathbf{g}}_c = \begin{cases} +1, & if \;\; \mathbf{g}_c \geq 0 \\ -1, & if \;\; \mathbf{g}_c < 0 \end{cases}. \tag{7}$$

A PostA-BM contains $K$ backward memories $\{\boldsymbol{\psi}^k\}_{k=1}^{K}$, respectively for the $K$ categories, where $\boldsymbol{\psi}^k \in \mathbb{R}^C$ stores the historical mean gradient after the activation TAU. For input data of each category $k$, the post-activation global feature gradient $\mathbf{g}^k$ within the network is computed through back-propagation, then binarized to $\tilde{\mathbf{g}}^k$ and updated in the historical mean $\boldsymbol{\psi}^k$ separately and in real time. To minimize space complexity, the update of $\boldsymbol{\psi}^k$ is also conducted in a moving mean manner, similar to how PreA-FM updates $\boldsymbol{\mu}^k$:

$$\boldsymbol{\psi}^{k,t} = (1-m) \cdot \boldsymbol{\psi}^{k,t-1} + m \cdot \tilde{\mathbf{g}}^{k,t}. \tag{8}$$

Based on the principle of gradient attribution, a positive average gradient of an activated channel indicates that this channel contributes positively to the prediction, whereas a negative average gradient indicates the opposite. The backward channel relevance criterion of each category denotes $\{\mathscr{R}^{k,B}\}_{k=1}^{K}$, where $B$ denotes the term "Backward" and $\mathscr{R}^{k,B} \in \mathbb{R}^C$ can be established as

$$\mathscr{R}_c^{k,B} = \begin{cases} 0, & if \;\; \boldsymbol{\psi}_c^k \geq 0 \\ 1, & if \;\; \boldsymbol{\psi}_c^k < 0 \end{cases}, \tag{9}$$

where the indicator "1" signifies potential irrelevant channels.

***Combining both the forward and backward propagation perspectives,*** the final criterion $\{\mathscr{R}^k\}_{k=1}^{K}$ for each category's channel relevance is the intersection (i.e., logical "and") of the forward criterion and the backward criterion:

$$\{\mathscr{R}^k\}_{k=1}^{K} = \{\mathscr{R}^{k,F} \cap \mathscr{R}^{k,B}\}_{k=1}^{K}. \tag{10}$$

Using a dual-perspective standard is potentially more accurate than making judgments solely from a single perspective.

### 3.3 Channel Denoising

The ultimate step, channel denoising, is conducted online on the activated feature map $\mathbf{A}$ under the guidance of the joint forward-backward criterion $\{\mathcal{R}^k\}_{k=1}^K$ for channel relevance.

Define the post-activation global feature value $\mathbf{a} \in \mathbb{R}^C$ to be the globally average pooled output of $\mathbf{A}$:

$$\mathbf{a} = \frac{1}{HW} \sum_{h=1}^H \sum_{w=1}^W \mathbf{A}_{h,w}. \tag{11}$$

Given an input data of category $k$, compute its activated feature value $\mathbf{a}^k$, then filter out irrelevant channels by $\mathcal{R}^k$:

$$\check{\mathbf{a}}^k = \mathbf{a}^k \odot \mathcal{R}^k, \tag{12}$$

where $\odot$ denotes the Hadamard Product. Finally, impose denoising on the filtered irrelevant channels by constructing a new loss item $\mathcal{L}_{ch}$ as follows:

$$\mathcal{L}_{ch} = \frac{\sum_{c=1}^C \left\| \check{\mathbf{a}}_c^k \right\|_2}{\sum_{c=1}^C \mathcal{R}_c^k}. \tag{13}$$

In this way, only the activation responses from channels with low correlation will be gradually suppressed, while the ones from other channels with high correlation are getting preserved.

### 3.4 Neural Network Learning

The proposed Dual-Perspective Activation (DPA) can replace the network's original ones. The channel denoising applies to the network's activated global feature. Networks that extract feature maps/sequences compute the global feature by taking the global average of the feature maps/sequences along the channels. Regarding some Transformer models that incorporate a class token, it is also possible to simply peel off the class token separately as the global feature vector while applying the corresponding denoising. Additionally, for Transformer models, the DPA is applied to each block, as Transformers excel in capturing global context throughout, while for CNN models, the proposed DPA is applied to the last block since high-level semantics only exist in deep representations [18]. The final loss $\mathcal{L}$ is expressed as

$$\mathcal{L} = \mathcal{L}_{task} + \lambda \cdot \frac{1}{N} \sum_{n=1}^N \mathcal{L}_{ch}^n, \tag{14}$$

where $\mathcal{L}_{task}$ is the primary loss for the specific task; for example, in the context of a standard classification task, $\mathcal{L}_{task}$ represents the cross-entropy loss. $N$ is the number of layers in the network that have channel denoising applied, and $\lambda$ is the balanced parameter.

## 4 Experimental Study

**Datasets.** We adopt six datasets, including four vision datasets: CIFAR-10 [19], CIFAR-100 [19], ImageNet-100 [20], and ImageNet-1K [20], and two non-vision datasets: DGraph [21] and 20News [22], to verify the effectiveness of the proposed DPA.

**Compared methods.** The proposed DPA is compared with different types of mainstream activation mechanisms mentioned in Related Work (§5.1), including Softplus [23], ELU [24], SELU [25], SiLU [26], ReLU [14], GELU [27], and GDN [28].

**Experimental settings.** The image size of CIFAR-{10,100} remains 32×32, while the images in ImageNet-{100,1K} are uniformly scaled to 224×224. To ensure the generality of the network, the activation threshold $\tau$ is uniformly set to 0. The momentum $m$ is empirically set to 0.9, and the balanced parameter $\lambda$ varies depending on networks and datasets. More detailed discussions on $\tau$, $m$, and $\lambda$ can be found in Appendix §A.1. All experiments use the same data augmentations provided by timm [29], AdamW optimizer with weight decay of 0.05, drop-path rate of 0.1, gradient clipping norm of 1.0, and cosine annealing learning rate scheduler with linear warm-up. All experiments are trained for 300 epochs from scratch. The automatic mixed precision training strategy is adopted to speed up the training. All other training settings, including batch size, learning rate, warm-up epochs, and so on, are kept identical throughout each set of comparative experiments. Note that the numerical results are the average under three different random seeds, and no pre-training is used.

Table 1: Top-1 accuracy (%) across the CIFAR-10, CIFAR-100, ImageNet-100, and ImageNet-1K datasets using the proposed DPA on Vision Transformer (ViT) and its variants.

| Top-1 Acc / % | | Softplus | ELU | SELU | SiLU | ReLU | GELU | GDN | DPA |
|---|---|---|---|---|---|---|---|---|---|
| CIFAR-10 | ViT-Tiny | 84.3 | 82.0 | 79.4 | 85.5 | 89.9 | 89.2 | 81.8 | **91.3** |
| | DeiT-Tiny | 84.7 | 81.4 | 79.9 | 86.6 | 89.6 | 89.2 | 83.0 | **91.5** |
| | CaiT-XXS | 82.5 | 80.7 | 78.4 | 86.6 | 89.4 | 88.7 | 80.0 | **91.4** |
| | PVT-Tiny | 90.6 | 89.3 | 85.4 | 92.5 | 93.0 | 92.8 | 82.8 | **93.8** |
| | TNT-Small | 88.3 | 85.4 | 83.7 | 90.5 | 90.8 | 91.1 | 85.1 | **92.4** |
| CIFAR-100 | ViT-Tiny | 62.4 | 60.0 | 57.5 | 65.5 | 65.7 | 65.4 | 59.4 | **70.5** |
| | DeiT-Tiny | 63.4 | 60.0 | 58.3 | 67.1 | 67.0 | 67.0 | 59.8 | **70.6** |
| | CaiT-XXS | 60.4 | 59.3 | 55.8 | 63.9 | 65.8 | 65.5 | 56.2 | **68.5** |
| | PVT-Tiny | 69.5 | 69.3 | 65.7 | 70.2 | 70.9 | 70.6 | 64.4 | **75.3** |
| | TNT-Small | 65.2 | 63.8 | 60.9 | 65.1 | 65.4 | 64.4 | 62.5 | **72.0** |
| ImageNet-100 | ViT-Tiny | 74.1 | 68.9 | 66.4 | 74.1 | 75.4 | 76.4 | 67.9 | **80.4** |
| | DeiT-Tiny | 75.3 | 69.4 | 67.0 | 75.1 | 75.6 | 74.6 | 66.3 | **81.0** |
| | CaiT-XXS | 70.9 | 69.1 | 65.9 | 76.1 | 76.0 | 76.7 | 69.5 | **80.4** |
| | PVT-Tiny | 79.5 | 77.1 | 76.1 | 79.5 | 81.9 | 81.4 | 75.8 | **85.2** |
| | TNT-Small | 78.9 | 79.3 | 76.4 | 77.6 | 79.9 | 77.2 | 76.9 | **85.6** |
| ImageNet-1K | ViT-Tiny | 70.0 | 64.2 | 63.1 | 66.9 | 70.9 | 70.4 | 65.2 | **72.2** |
| | DeiT-Tiny | 71.9 | 67.9 | 66.2 | 72.0 | 73.2 | 73.0 | 66.4 | **73.4** |
| | CaiT-XXS | 70.3 | 68.1 | 66.7 | 73.2 | 74.0 | 73.6 | 66.1 | **75.0** |
| | PVT-Tiny | 71.5 | 69.2 | 68.5 | 72.8 | 73.7 | 73.5 | 66.5 | **75.2** |
| | TNT-Small | 72.0 | 70.7 | 70.3 | 71.5 | 73.4 | 73.3 | 68.2 | **77.8** |

Table 2: Top-1 accuracy (%) across the CIFAR-10, CIFAR-100, ImageNet-100, and ImageNet-1K datasets using the proposed DPA on various CNN architectures.

| Top-1 Acc / % | | Softplus | ELU | SELU | SiLU | ReLU | GELU | GDN | DPA |
|---|---|---|---|---|---|---|---|---|---|
| CIFAR-10 | AlexNet | 85.6 | 86.1 | 85.7 | 86.0 | 86.0 | 85.8 | 85.4 | **86.4** |
| | VGG-11 | 91.3 | 92.0 | 91.5 | 91.9 | **92.2** | 91.9 | 91.1 | **92.2** |
| | MobileNet | 87.4 | 87.7 | 87.2 | **87.8** | 87.4 | 87.4 | 87.0 | **87.8** |
| | ShuffleNet | 89.2 | 89.0 | 88.9 | 89.3 | 89.4 | 89.3 | 88.5 | **89.8** |
| | ResNet-18 | 94.6 | 94.7 | 94.6 | **95.1** | 95.0 | 94.9 | 94.0 | **95.1** |
| CIFAR-100 | AlexNet | 57.6 | 58.4 | 58.1 | 58.1 | 57.2 | 57.4 | 56.8 | **58.5** |
| | VGG-11 | 69.6 | 69.9 | 69.7 | 69.9 | 70.2 | 70.0 | 70.1 | **70.3** |
| | MobileNet | 65.4 | 65.5 | 65.6 | 65.2 | 66.0 | 65.4 | 64.8 | **67.2** |
| | ShuffleNet | 66.2 | 66.1 | 65.9 | 66.3 | 66.3 | 66.2 | 65.6 | **66.8** |
| | ResNet-18 | 75.5 | 75.7 | 75.6 | 76.1 | 75.7 | 75.6 | 74.3 | **76.8** |
| ImageNet-100 | AlexNet | 75.7 | 76.0 | 75.7 | 76.6 | 76.3 | 76.3 | 75.5 | **76.9** |
| | VGG-11 | 87.0 | 87.3 | 87.6 | 87.8 | 87.7 | 87.5 | 86.7 | **88.4** |
| | MobileNet | 80.6 | 79.3 | 79.2 | 80.1 | 80.6 | 80.5 | 78.7 | **81.7** |
| | ShuffleNet | 80.9 | 80.9 | 80.4 | 81.7 | 81.6 | 81.6 | 80.0 | **81.9** |
| | ResNet-18 | 84.6 | 84.4 | 84.1 | 84.9 | 84.9 | 84.7 | 83.5 | **85.7** |
| ImageNet-1K | AlexNet | 56.1 | 56.3 | 56.1 | 56.4 | 56.5 | 56.4 | 55.6 | **57.5** |
| | VGG-11 | 68.4 | 68.2 | 67.8 | 69.0 | 69.0 | 69.1 | 68.1 | **69.7** |
| | MobileNet | 67.2 | 66.7 | 67.1 | 67.4 | 68.1 | 68.2 | 66.3 | **68.9** |
| | ShuffleNet | 68.5 | 68.3 | 68.4 | 69.1 | 69.0 | 68.9 | 68.0 | **69.5** |
| | ResNet-18 | 69.3 | 69.4 | 68.9 | 69.7 | 69.7 | 69.4 | 68.3 | **70.3** |

## 4.1 DPA on ViTs

The proposed DPA mechanism can be incorporated into popular Vision Transformer (ViT) and its variants. Table 1 shows the top-1 accuracy (%) across CIFAR-{10,100} and ImageNet-{100,1K} using the proposed DPA on five different ViT architectures: ViT [30], DeiT [31], CaiT [32] PVT [33], and TNT [34]. The proposed DPA can replace all the existing activations in each block. The results consistently illustrate that the proposed DPA mechanism outperforms the baselines.

Table 3: Ablation study of the proposed DPA on CIFAR-100. $\text{DPA}_F$, $\text{DPA}_B$, $\text{DPA}_{F \cup B}$ and $\text{DPA}_{F \cap B}$ denote DPA with the channel relevance criterion in the form of forward only, backward only, forward-backward union, and forward-backward intersection, respectively. $\text{DPA}_{all}$ denotes denoising all channels indiscriminately.

| Top-1 Acc / % | ReLU | GELU | $\text{DPA}_{all}$ | $\text{DPA}_F$ | $\text{DPA}_B$ | $\text{DPA}_{F \cup B}$ | $\text{DPA}_{F \cap B}$ |
|---|---|---|---|---|---|---|---|
| ViT-Tiny | 65.7 | 65.4 | 65.8 | 68.9 | 67.9 | 67.6 | **70.5** |
| ResNet-18 | 75.7 | 75.6 | 75.7 | 76.5 | 76.2 | 76.0 | **76.8** |

Table 4: Computational costs during training and inference regarding "GPU Memory (GiB)" and "Latency (s)" for networks that utilize the proposed DPA, in comparison to networks utilizing other activation counterparts. The networks were fed 224×224-pixel images with a batch size of 1024 on an NVIDIA A40 GPU. "Latency" refers to the average time it takes for a network to process a batch of data.

| Computational Costs | ViT-Tiny | | | ResNet-18 | | |
|---|---|---|---|---|---|---|
| | ReLU | GELU | DPA | ReLU | GELU | DPA |
| Training GPU Memory / GiB | 33.09 | 37.14 | 33.11 | 27.06 | 30.38 | 27.08 |
| Inference GPU Memory / GiB | 6.26 | 6.26 | 6.26 | 11.85 | 11.86 | 11.85 |
| Training Latency / s | 0.71 | 0.83 | 0.89 | 0.57 | 0.65 | 0.62 |
| Inference Latency / s | 0.39 | 0.42 | 0.39 | 0.39 | 0.39 | 0.39 |

Table 5: The generalization performance of DPA on node classification and text classification tasks.

| Node Classification | | | Text Classification | | |
|---|---|---|---|---|---|
| AUC / % | GCN | GraphSAGE | Top-1 Acc / % | TextGCN | BERT |
| ReLU | 72.5 | 75.1 | ReLU | 86.2 | 86.7 |
| GELU | 71.9 | 74.4 | GELU | 86.1 | 86.9 |
| DPA | **73.7** | **76.6** | DPA | **86.9** | **87.8** |

## 4.2 DPA on CNNs

The proposed DPA is also evaluated on various mainstream CNNs, including AlexNet [35], VGG [36], MobileNet [37], ShuffleNet(V2) [38], and ResNet [39]. The proposed DPA replaces the original activations in the last block since previous works have shown that high-level semantics in CNNs only exist in deep representations [18]. The results in Table 2 highlight the versatility and robustness of the proposed DPA in handling diverse CNN architectures and datasets.

## 4.3 Ablation Study

The ablation study mainly focuses on different combinations of the forward and backward criteria. Table 3 presents the results, showing that the performance with the dual-perspective criterion ($\text{DPA}_{F \cap B}$) is better than using a single-perspective criterion ($\text{DPA}_F$ or $\text{DPA}_B$). Furthermore, the intersection of the dual perspectives ($\text{DPA}_{F \cap B}$) yields better results than the union of both ($\text{DPA}_{F \cup B}$), as the intersection of multiple perspectives can reduce misjudgments. Additionally, if denoise all channels indiscriminately ($\text{DPA}_{all}$), there will be no performance improvement.

## 4.4 Computational Costs

The proposed DPA introduces no extra parameters in any of its components. The two memory units (PreA-FM and PostA-BM) only work during the training phase, and in the inference phase, only TAU needs to be involved. The numerical computational costs during training and inference regarding "GPU Memory (GiB)" and "Latency (s)" (the average time it takes for a network to process a batch of data) are shown in Table 4. Notably, the activation counterparts used in original networks should be implemented manually as our DPA does. Utilizing the counterparts directly from pre-made libraries (like torch.nn) can result in unfair comparisons due to their high optimization at the low level. Table 4 indicates that the GPU overhead required by DPA during training is negligible, and the speed of DPA is on par with other counterparts during the inference stage.

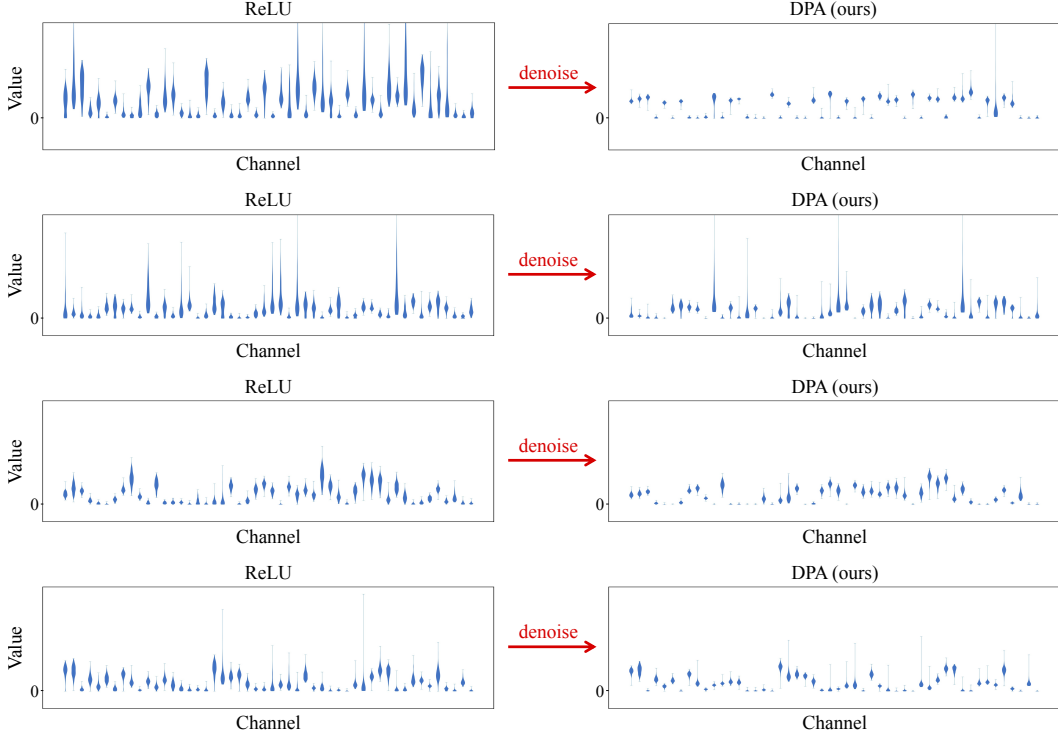

Figure 4: Distributions of channel activation values in ViT-Tiny's last block are recorded by feeding samples from a given category in CIFAR-100. The proposed DPA is compared with ReLU. The horizontal axis represents the channel index, the vertical axis represents the channel activation value, and the area represents the value density. In the figure, each row displays the changes in response distributions for a specific category when transitioning from ReLU to the proposed DPA. The distributions of the first 50 channels are shown.

## 4.5 Generalization to Other Tasks

The proposed DPA can also perform various other tasks or domains, including non-vision tasks like node and text classification. For node classification, the DGraph dataset [21] is employed, and GCN [40] and GraphSAGE [41] are chosen models. For text classification, the 20News dataset [22] is employed, and BERT [42] and TextGCN [43] are chosen models. The results are shown in Table 5.

## 4.6 Activation Response Visualization

Figure 4 displays the change in channel-wise activation distributions of a specific category in the last block of ViT-Tiny after using ReLU and the proposed DPA. With the typical ReLU activation, the presented responses are active on almost every channel. Some irrelevant channels that should not be activated are mistakenly activated. After using the proposed DPA, the activation responses become sparser, implying that most irrelevant channels are suppressed. Moreover, the suppression of irrelevant channels leads to enhanced focus in the responses from other channels. These phenomena imply that the DPA facilitates the extraction of key features from input data with higher precision, which reduces the network's learning difficulty and improves its interpretability.

# 5 Related Work

## 5.1 Forward Activation Mechanism

The activation mechanism [15] plays a pivotal role in artificial neural networks as it defines how neurons respond to input signals, convert them into output signals, and transmit them to the subsequent layer. Each activation mechanism varies in mathematical properties and nonlinear forms, allowing neurons to simulate different neural phenomena, including excitation, inhibition, and modulation. Mathematically, activation mechanisms are categorized into different types, including logistic Sigmoid

and Tanh variants [9], Rectified Linear Unit variants [14], Exponential Linear Unit variants [24], Softplus variants [23], probabilistic variants [27], and others [26, 28].

In the widely used form of activation [9, 14, 23, 24, 27], irrelevant features are suppressed, and relevant features gain amplified influence according to the response rule of the neuron. Furthermore, some activation mechanisms [9, 14, 23] can effectively achieve data sparsity, diminish redundant information, and enable better feature distinction. Additionally, the activation mechanisms, such as ELU [24] and SiLU [26] mentioned above, and others [44, 45], contain learnable parameters. Parameters in [44] can adapt to various data distributions, avoiding gradient vanishing and explosion, thereby enhancing the convergence speed and precision of ANNs. However, these extra parameters can only uniformly influence the response strength for all inputs. Parameters in [45] allow the original activation to adjust its response condition according to different external inputs. The inclusion of these extra parameters offer potential for the activation to eliminate irrelevant noise. Nevertheless, it is important to note that this is not a targeted design, since these extra parameters are completely adaptively learned without supervision signals, rather than specifically identifying irrelevant channels and selectively applying channel denoising.

## 5.2 Backward Gradient Attribution

The gradient attribution [46] includes a set of methods utilized to explain the predictions made by ANNs. It is generally believed that the gradient of the neural network output w.r.t. the input can indicate the importance or relevance of the input.

Baehrens et al. [47] were pioneers in using the first-order derivative of the predicted class w.r.t. the input to elucidate the local decisions made by nonlinear classification algorithms. Simonyan et al. [48] employed the gradient to compute a class saliency map that is specific to a given image and class. Bach et al. [49] further developed the Layer-wise Relevance Propagation (LRP), which attributes by propagating gradients layer by layer. Shrikumar et al. [50] proposed DeepLIFT, which uses backpropagation to identify each neuron's contribution in a neural network toward the output prediction. Sundararajan et al. [51] introduced the Integrated Gradients algorithm to counter gradient saturation, which calculates gradient computation as the path integral of the first-order derivative of the output w.r.t. the input features along a straight path from a baseline to the input.

Additionally, the gradient-based Class Activation Mapping (CAM) family [52–54] has become widely popular for its computational efficiency, requiring no structural changes or re-training. The most classical one is Grad-CAM [52], which can be used to explain activations in any layer of a deep network by computing the importance (relevance score) for each channel of the activated feature map and performing a weighted sum. The relevance score of a channel is derived by taking the average gradient of the predicted class score w.r.t. the activated feature map in that channel, which motivates us to utilize similar principles when designing criteria for channel relevance in our study.

## 6 Conclusion

It is observed in Artificial Neural Networks (ANNs) that each category is only associated with sparse and specific channels, however, current activation mechanisms often struggle to suppress the signals from other irrelevant channels, which negatively impacts the network's final decision. To alleviate such noise interference, a novel end-to-end trainable Dual-Perspective Activation (DPA) mechanism is proposed. DPA is guided by a joint criterion established online from both forward and backward propagation perspectives, aiming to efficiently identify irrelevant channels and apply channel denoising while preserving activation responses from relevant channels. Extensive experiments showcase that DPA is compatible with various mainstream ANN architectures, can achieve sparser neural representations and outperform other activation counterparts, demonstrating its effectiveness and versatility. Additionally, DPA is parameter-free and offers fast inference speed on par with other activation counterparts, indicating its potential for practical application in real-world scenarios.

## Acknowledgments

This work is supported by the Zhejiang Province "JianBingLingYan+X" Research and Development Plan (2024C01114).

# References

[1] Yann LeCun, Yoshua Bengio, and Geoffrey Hinton. Deep learning. *Nature*, 521(7553):436–444, 2015.

[2] Athanasios Voulodimos, Nikolaos Doulamis, Anastasios Doulamis, Eftychios Protopapadakis, et al. Deep learning for computer vision: A brief review. *Computational intelligence and neuroscience*, 2018, 2018.

[3] Zonghan Wu, Shirui Pan, Fengwen Chen, Guodong Long, Chengqi Zhang, and S Yu Philip. A comprehensive survey on graph neural networks. *IEEE transactions on neural networks and learning systems*, 32(1):4–24, 2020.

[4] Shervin Minaee, Nal Kalchbrenner, Erik Cambria, Narjes Nikzad, Meysam Chenaghlu, and Jianfeng Gao. Deep learning–based text classification: a comprehensive review. *ACM computing surveys*, 54(3):1–40, 2021.

[5] Jingxuan He, Lechao Cheng, Chaowei Fang, Zunlei Feng, Tingting Mu, and Mingli Song. Progressive feature self-reinforcement for weakly supervised semantic segmentation. In *Proceedings of the AAAI conference on artificial intelligence*, volume 38, pages 2085–2093, 2024.

[6] Yuzhu Wang, Lechao Cheng, Manni Duan, Yongheng Wang, Zunlei Feng, and Shu Kong. Improving knowledge distillation via regularizing feature norm and direction. *arXiv preprint arXiv:2305.17007*, 2023.

[7] Warren S McCulloch and Walter Pitts. A logical calculus of the ideas immanent in nervous activity. *The bulletin of mathematical biophysics*, 5:115–133, 1943.

[8] James L McClelland, David E Rumelhart, PDP Research Group, et al. *Parallel Distributed Processing, Volume 2: Explorations in the Microstructure of Cognition: Psychological and Biological Models*, volume 2. MIT press, 1987.

[9] Yann LeCun, Léon Bottou, Yoshua Bengio, and Patrick Haffner. Gradient-based learning applied to document recognition. *Proceedings of the IEEE*, 86(11):2278–2324, 1998.

[10] Kai Han, Yunhe Wang, Hanting Chen, Xinghao Chen, Jianyuan Guo, Zhenhua Liu, Yehui Tang, An Xiao, Chunjing Xu, Yixing Xu, et al. A survey on vision transformer. *IEEE transactions on pattern analysis and machine intelligence*, 45(1):87–110, 2022.

[11] David Attwell and Simon B Laughlin. An energy budget for signaling in the grey matter of the brain. *Journal of cerebral blood flow & metabolism*, 21(10):1133–1145, 2001.

[12] Peter Lennie. The cost of cortical computation. *Current biology*, 13(6):493–497, 2003.

[13] Bruno A Olshausen and David J Field. Emergence of simple-cell receptive field properties by learning a sparse code for natural images. *Nature*, 381(6583):607–609, 1996.

[14] Xavier Glorot, Antoine Bordes, and Yoshua Bengio. Deep sparse rectifier neural networks. In *Proceedings of the fourteenth international conference on artificial intelligence and statistics*, pages 315–323. JMLR workshop and conference proceedings, 2011.

[15] Shiv Ram Dubey, Satish Kumar Singh, and Bidyut Baran Chaudhuri. Activation functions in deep learning: A comprehensive survey and benchmark. *Neurocomputing*, 503:92–108, 2022.

[16] Zunlei Feng, Jiacong Hu, Sai Wu, Xiaotian Yu, Jie Song, and Mingli Song. Model doctor: A simple gradient aggregation strategy for diagnosing and treating cnn classifiers. In *Proceedings of the AAAI conference on artificial intelligence*, volume 36, pages 616–624, 2022.

[17] Alan L Hodgkin and Andrew F Huxley. A quantitative description of membrane current and its application to conduction and excitation in nerve. *The Journal of physiology*, 117(4):500, 1952.

[18] Maithra Raghu, Thomas Unterthiner, Simon Kornblith, Chiyuan Zhang, and Alexey Dosovitskiy. Do vision transformers see like convolutional neural networks? *Advances in neural information processing systems*, 34:12116–12128, 2021.

[19] Alex Krizhevsky, Geoffrey Hinton, et al. Learning multiple layers of features from tiny images. 2009.

[20] Jia Deng, Wei Dong, Richard Socher, Li-Jia Li, Kai Li, and Li Fei-Fei. Imagenet: A large-scale hierarchical image database. In *IEEE conference on computer vision and pattern recognition*, pages 248–255. IEEE, 2009.

[21] Xuanwen Huang, Yang Yang, Yang Wang, Chunping Wang, Zhisheng Zhang, Jiarong Xu, Lei Chen, and Michalis Vazirgiannis. Dgraph: A large-scale financial dataset for graph anomaly detection. *Advances in Neural Information Processing Systems*, 35:22765–22777, 2022.

[22] The 20 newsgroups dataset. http://qwone.com/ jason/20Newsgroups/., 2021.

[23] Charles Dugas, Yoshua Bengio, François Bélisle, Claude Nadeau, and René Garcia. Incorporating second-order functional knowledge for better option pricing. *Advances in neural information processing systems*, 13, 2000.

[24] Djork-Arné Clevert, Thomas Unterthiner, and Sepp Hochreiter. Fast and accurate deep network learning by exponential linear units (elus). *arXiv preprint arXiv:1511.07289*, 2015.

[25] Günter Klambauer, Thomas Unterthiner, Andreas Mayr, and Sepp Hochreiter. Self-normalizing neural networks. *Advances in neural information processing systems*, 30, 2017.

[26] Prajit Ramachandran, Barret Zoph, and Quoc V Le. Searching for activation functions. *arXiv preprint arXiv:1710.05941*, 2017.

[27] Dan Hendrycks and Kevin Gimpel. Gaussian error linear units (gelus). *arXiv preprint arXiv:1606.08415*, 2016.

[28] Johannes Ballé, Valero Laparra, and Eero P Simoncelli. Density modeling of images using a generalized normalization transformation. *arXiv preprint arXiv:1511.06281*, 2015.

[29] Ross Wightman. Pytorch image models. https://github.com/rwightman/pytorch-image-models, 2019.

[30] Alexey Dosovitskiy, Lucas Beyer, Alexander Kolesnikov, Dirk Weissenborn, Xiaohua Zhai, Thomas Unterthiner, Mostafa Dehghani, Matthias Minderer, Georg Heigold, Sylvain Gelly, et al. An image is worth 16x16 words: Transformers for image recognition at scale. *arXiv preprint arXiv:2010.11929*, 2020.

[31] Hugo Touvron, Matthieu Cord, Matthijs Douze, Francisco Massa, Alexandre Sablayrolles, and Hervé Jégou. Training data-efficient image transformers & distillation through attention. In *International conference on machine learning*, pages 10347–10357. PMLR, 2021.

[32] Hugo Touvron, Matthieu Cord, Alexandre Sablayrolles, Gabriel Synnaeve, and Hervé Jégou. Going deeper with image transformers. In *Proceedings of the IEEE/CVF international conference on computer vision*, pages 32–42, 2021.

[33] Wenhai Wang, Enze Xie, Xiang Li, Deng-Ping Fan, Kaitao Song, Ding Liang, Tong Lu, Ping Luo, and Ling Shao. Pyramid vision transformer: A versatile backbone for dense prediction without convolutions. In *Proceedings of the IEEE/CVF international conference on computer vision*, pages 568–578, 2021.

[34] Kai Han, An Xiao, Enhua Wu, Jianyuan Guo, Chunjing Xu, and Yunhe Wang. Transformer in transformer. *Advances in neural information processing systems*, 34:15908–15919, 2021.

[35] Alex Krizhevsky, Ilya Sutskever, and Geoffrey E Hinton. Imagenet classification with deep convolutional neural networks. *Communications of the ACM*, 60(6):84–90, 2017.

[36] Karen Simonyan and Andrew Zisserman. Very deep convolutional networks for large-scale image recognition. *arXiv preprint arXiv:1409.1556*, 2014.

[37] Andrew G Howard, Menglong Zhu, Bo Chen, Dmitry Kalenichenko, Weijun Wang, Tobias Weyand, Marco Andreetto, and Hartwig Adam. Mobilenets: Efficient convolutional neural networks for mobile vision applications. *arXiv preprint arXiv:1704.04861*, 2017.

[38] Ningning Ma, Xiangyu Zhang, Hai-Tao Zheng, and Jian Sun. Shufflenet v2: Practical guidelines for efficient cnn architecture design. In *Proceedings of the european conference on computer vision*, pages 116–131, 2018.

[39] Kaiming He, Xiangyu Zhang, Shaoqing Ren, and Jian Sun. Deep residual learning for image recognition. In *Proceedings of the IEEE conference on computer vision and pattern recognition*, pages 770–778, 2016.

[40] Thomas Kipf and Max Welling. Semi-supervised classification with graph convolutional networks. *ArXiv*, abs/1609.02907, 2016.

[41] Will Hamilton, Zhitao Ying, and Jure Leskovec. Inductive representation learning on large graphs. *Advances in neural information processing systems*, 30, 2017.

[42] Jacob Devlin, Ming-Wei Chang, Kenton Lee, and Kristina Toutanova. Bert: Pre-training of deep bidirectional transformers for language understanding. *arXiv preprint arXiv:1810.04805*, 2018.

[43] Liang Yao, Chengsheng Mao, and Yuan Luo. Graph convolutional networks for text classification. In *Proceedings of the AAAI conference on artificial intelligence*, volume 33, pages 7370–7377, 2019.

[44] Aizhu Liu, Haigen Hu, Tian Qiu, Qianwei Zhou, Qiu Guan, and Xiaoxin Li. Exploring optimal adaptive activation functions for various tasks. In *IEEE international conference on bioinformatics and biomedicine (BIBM)*, pages 2290–2297. IEEE, 2020.

[45] Tian Qiu, Wenxiang Xu, Lin Chen, Linyun Zhou, Zunlei Feng, and Mingli Song. Dynamic neural response tuning. In *The twelfth international conference on learning representations*, 2024.

[46] Marco Ancona, Enea Ceolini, Cengiz Öztireli, and Markus Gross. Gradient-based attribution methods. *Explainable AI: interpreting, explaining and visualizing deep learning*, pages 169–191, 2019.

[47] David Baehrens, Timon Schroeter, Stefan Harmeling, Motoaki Kawanabe, Katja Hansen, and Klaus-Robert Müller. How to explain individual classification decisions. *The journal of machine learning research*, 11:1803–1831, 2010.

[48] Karen Simonyan, Andrea Vedaldi, and Andrew Zisserman. Deep inside convolutional networks: Visualising image classification models and saliency maps. *arXiv preprint arXiv:1312.6034*, 2013.

[49] Sebastian Bach, Alexander Binder, Grégoire Montavon, Frederick Klauschen, Klaus-Robert Müller, and Wojciech Samek. On pixel-wise explanations for non-linear classifier decisions by layer-wise relevance propagation. *PloS one*, 10(7):e0130140, 2015.

[50] Avanti Shrikumar, Peyton Greenside, and Anshul Kundaje. Learning important features through propagating activation differences. In *International conference on machine learning*, pages 3145–3153, 2017.

[51] Mukund Sundararajan, Ankur Taly, and Qiqi Yan. Axiomatic attribution for deep networks. In *International conference on machine learning*, pages 3319–3328. PMLR, 2017.

[52] Ramprasaath R Selvaraju, Michael Cogswell, Abhishek Das, Ramakrishna Vedantam, Devi Parikh, and Dhruv Batra. Grad-cam: Visual explanations from deep networks via gradient-based localization. In *Proceedings of the IEEE international conference on computer vision*, pages 618–626, 2017.

[53] Aditya Chattopadhay, Anirban Sarkar, Prantik Howlader, and Vineeth N Balasubramanian. Grad-cam++: Generalized gradient-based visual explanations for deep convolutional networks. In *IEEE winter conference on applications of computer vision*, pages 839–847. IEEE, 2018.

[54] Daniel Omeiza, Skyler Speakman, Celia Cintas, and Komminist Weldermariam. Smooth grad-cam++: An enhanced inference level visualization technique for deep convolutional neural network models. *arXiv preprint arXiv:1908.01224*, 2019.

# A  Appendix

## A.1  Hyperparameter Impact Analysis

**Momentum $m$ for updating the moving mean μ.** Theoretically, a high value of $m$ risks making the mean value unstable, and conversely, a low value of $m$ smoothens the mean value update, but it may cause the mean value to lag behind. As illustrated in Figure 5, the network's performance is not sensitive to the $m$ between around 0.2 and 0.99, from which $m$=0.9 performs the best, and extremely small $m$ values lead to negative effects. Therefore, for the rest of the experiments presented in the paper, we empirically set the $m$ to 0.9 without too much consideration.

**Balanced parameter $\lambda$ for the channel loss $\mathcal{L}_{ch}$.** The optimal balanced parameter $\lambda$ for $\mathcal{L}_{ch}$ is specific to individual tasks. The relationship between $\lambda$ and the accuracy on CIFAR-100 with the ViT-Tiny model is depicted in Figure 6. In this case, the optimal $\lambda$ is roughly 5, and too large $\lambda$ can result in negative side effects. For other trials, we found the optimal $\lambda$ to be 5 when training DeiT-Tiny and TNT-Small on CIFAR-100 and the optimal $\lambda$ to be 1 when training ViT-Tiny on CIFAR-10. Searching the optimal parameters for each task can be time-consuming, but one thing is for sure: smaller $\lambda$ values do not hurt accuracy. Therefore, for the majority of our experiments, we set the default value of $\lambda$ to 1.

**Activation threshold $\tau$ for the Threshold Activation Unit (TAU).** To achieve preliminary sparsity, only input signals that exceed the threshold $\tau$ are activated, while those below the threshold are suppressed to zero. Therefore, $\tau$ is a non-negative value. Figure 7 demonstrates that as $\tau$ increases, the performance decreases. Possible reasons could be the influence of weight initialization and feature normalization operations. Typically, weights are initialized using a distribution with a mean of zero, and normalization techniques such as layer normalization and batch normalization are used to make the feature distribution centered around zero (by subtracting the feature mean) to eliminate shifts in data covariates. Under this circumstance, $\tau = 0$ becomes the optimal activation threshold. Additionally, it may be feasible to modify the strategy for weight initialization and feature normalization to achieve optimal effects when considering a positive $\tau$.

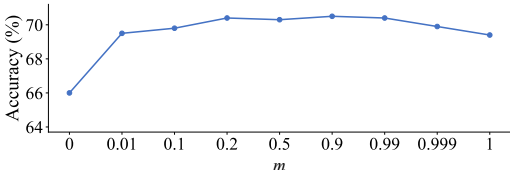

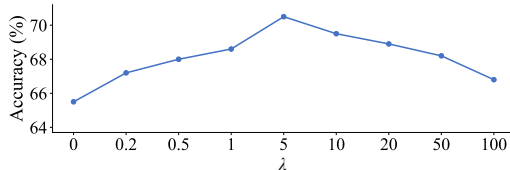

Figure 5: Top-1 accuracy (%) w.r.t. the momentum $m$ for updating the moving mean μ when training on CIFAR-100 with ViT-Tiny.

Figure 6: Top-1 accuracy (%) w.r.t. the balanced parameter $\lambda$ for the channel loss $\mathcal{L}_{ch}$ when training on CIFAR-100 with ViT-Tiny.

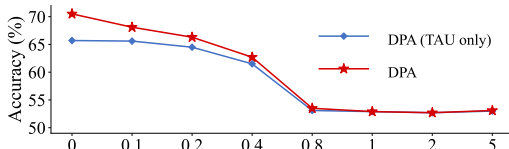

Figure 7: Top-1 accuracy (%) w.r.t. the activation threshold $\tau$ for the Threshold Activation Unit (TAU) when training on CIFAR-100 with ViT-Tiny.

## A.2  Overfitting Test

Deep neural networks are susceptible to overfitting when the available data is insufficient. Therefore, the network requires robust support from data augmentations as illustrated in this study. We assess the performance of ViT-Tiny with ReLU and the proposed DPA on the CIFAR-10 dataset when using weak data augmentations consisting solely of "random horizontal flipping" and "normalization". Figure 8 showcases the loss and accuracy curves. Specifically, the "val" loss curve provides further evidence that DPA effectively mitigates overfitting by reducing noise interference from irrelevant channels and promoting sparsity on the representation level.

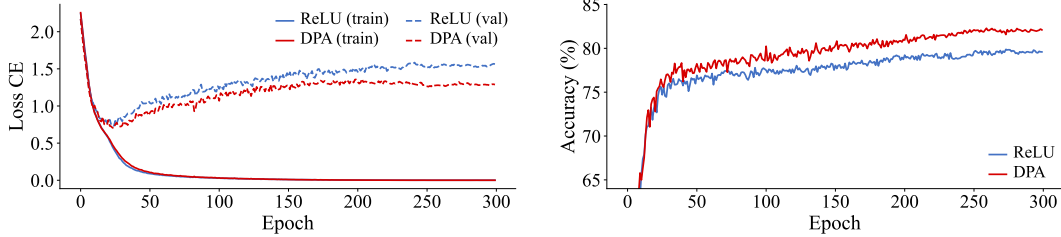

Figure 8: The performance of ViT-Tiny with ReLU and the proposed DPA on the CIFAR-10 dataset using weak data augmentations comprising only "random horizontal flipping" and "normalization".

### A.3 Differences between Channel Denoising and Channel Pruning

**(a)** Explanation of "channel denoising": i) We regard the responses from potential irrelevant channels as channel noise (see Figure 3(a)). ii) Also, each channel tends to exhibit response distributions with high variance, which is also a symbol of channel noise (see Figure 3(a)). Therefore, the proposed DPA aims to denoise these channels (i & ii), thus achieving **sparser representations** and **remarkable accuracy improvements**. Figure 4 shows the denoising effect where the responses (not the weights) from irrelevant channels are suppressed (not forcibly set to zero), and the ones from relevant channels become more focused.

**(b)** The proposed channel denoising differs from the channel pruning as:

- The target of channel pruning is the channel weights $\{\mathbf{W}_c\}_{c=1}^{C}$, while the target of channel denoising is the channel responses $f(x; \{\mathbf{W}_c\}_{c=1}^{C})$. Specifically, for the $c$-th channel, given an input sample $x$ and the channel weight $\mathbf{W}_c$, it generates a channel response $f(x; \mathbf{W}_c)$. Channel pruning operates by setting **the weights of certain channels** to zero, resulting in no responses on pruned channels for samples of any category (This is why channel pruning typically leads to a decrease in model accuracy.). In contrast, channel denoising operates by suppressing **the responses of certain channels** to zero, in which case, the channel weights are not necessarily zero, and the suppressed channels can vary for samples of different categories.

- Channel pruning needs post-processing to remove irrelevant channel weights and some need further fine-tuning, while channel denoising is conveniently trained end-to-end from scratch and does not require any post-processing or fine-tuning.

- The objective of channel pruning is trying to reduce computation and storage requirements without sacrificing accuracy, while the objective of channel denoising is trying to improve accuracy without increasing computational overhead.

### A.4 Limitation and Discussion

**Performance.** The proposed DPA still has a gap to achieve the ideal performance as *Observation 2* shows. This limitation can be attributed to several factors. Firstly, the confirmatory experiment in *Observation 2* assumes the category label is always known, but in applications, the category label is unknown during the testing phase. Therefore, to address this challenge, we introduced channel denoising during training, allowing the network to learn how to reduce the response from irrelevant channels in the testing stage when the category label is unknown. Through this approach, we expect to approximate the way of manually removing irrelevant channels in *Observation 2* as much as possible. However, the joint criterion devised for identifying irrelevant channels is not always precise, highly depending on the historical context of each training moment, and some categories that share similarities at the representation level could add to the difficulty of precise denoising while still preserving category discrimination. At present, we are actively researching these areas and are confident that it will achieve more promising results.

**Tasks.** In certain scenarios, e.g., the box regression part of object detection, the proposed DPA may not be applicable. This is because DPA is associated with categories, whereas box regression is not. Despite this, the generalization ability of DPA is already commendable, as it can perform various other tasks or domains, including image classification and non-vision tasks like node classification and text classification, whose generalization has been verified in the presented experiments.

